# Using body-anchored priors for identifying actions in single images

**Leonid Karlinsky**　　　**Michael Dinerstein**　　　**Shimon Ullman**
Department of Computer Science
Weizmann Institute of Science
Rehovot 76100, Israel
`{leonid.karlinsky, michael.dinerstein, shimon.ullman}` @weizmann.ac.il

## Abstract

This paper presents an approach to the visual recognition of human actions using only single images as input. The task is easy for humans but difficult for current approaches to object recognition, because instances of different actions may be similar in terms of body pose, and often require detailed examination of relations between participating objects and body parts in order to be recognized. The proposed approach applies a two-stage interpretation procedure to each training and test image. The first stage produces accurate detection of the relevant body parts of the actor, forming a prior for the local evidence needed to be considered for identifying the action. The second stage extracts features that are anchored to the detected body parts, and uses these features and their feature-to-part relations in order to recognize the action. The body anchored priors we propose apply to a large range of human actions. These priors allow focusing on the relevant regions and relations, thereby significantly simplifying the learning process and increasing recognition performance.

## 1   Introduction

This paper deals with the problem of recognizing transitive actions in single images. A *transitive action* is often described by a transitive verb and involves a number of components, or thematic roles [1], including an actor, a tool, and in some cases a recipient of the action. Simple examples are drinking from a glass, talking on the phone, eating from a plate with a spoon, or brushing teeth. Transitive actions are characterized visually by the posture of the actor, the tool she/he is holding, the type of grasping, and the presence of the action recipient. In many cases, such actions can be readily identified by human observers from only a single image (see figure 1a). We will consider below the problem of static action recognition (SAR for short) from a single image, without using motion information that is exploited by approaches dealing with dynamic action recognition in video sequences, such as [2]. The problem is of interest first, because in a short observation interval, the use of motion information for identifying an action (e.g. talking on the phone) may be limited. Second, as a natural human capacity, it is of interest for both cognitive and brain studies. Several studies [3, 4, 5, 6] have shown evidence for the presence of SAR related mechanisms in both the ventral and dorsal areas of the visual cortex, and computational modeling of SAR may shed new light on these mechanisms. Unlike the more common task of detecting individual objects such as faces and cars, SAR depends on detecting object configurations. Different actions may involve the same type of objects (eg. person, phone) but appearing in different configurations (answering, dialing), sometimes differing in subtle details, making their identification difficult compared with individual object recognition.

Only a few approaches to date have dealt with the SAR problem. [7] studied the recognition of sports actions using the pose of the actor. [8] used scene interpretation in terms of objects and

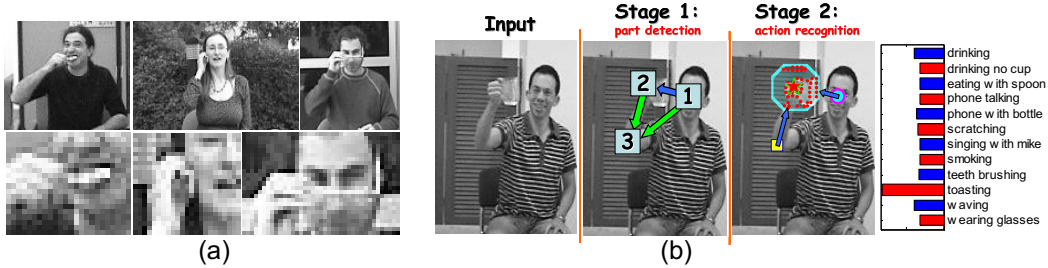

Figure 1: (a) Examples of similar transitive actions identifiable by humans from single images (brushing teeth, talking on a cell phone and wearing glasses). (b) Illustration of a single run of the proposed two-stage approach. In the first stage the parts are detected in the face→hand→elbow order. In the second stage we apply both action learning and action recognition using the configuration of the detected parts and the features anchored to the hand region; the bar graph on the right shows relative log-posterior estimates for the different actions.

their relative configuration to distinguish between different sporting events such as badminton and sailing. [9] recognized static intransitive actions such as walking and jumping based on a human body pose represented by a variant of the HOG descriptor. [10] discriminated between playing and not playing musical instruments using a star-like model. The most detailed static schemes to date [11, 12] recognized static transitive sports actions, such as the tennis forehand and the volleyball smash. [11] used a full body mask, bag of features for describing scene context, and the detection of the objects relevant for the action, such as bats, balls, etc., while [12] learned joint models of body pose and objects specific to each action. [11] used GrabCut [13] to extract the body mask, and both [11] and [12] performed fully supervised training for the a priori known relevant objects and scenes.

In this paper we consider the task of differentiating between similar types of transitive actions, such as smoking a cigarette, drinking from a cup, eating from a cup with a spoon, talking on the phone, etc., given only a single image as input. The similarity between the body poses in such actions creates a difficulty for approaches that rely on pose analysis [7, 9, 11]. The relevant differences between similar actions in terms of the actor body configuration can be at a fine level of detail. Therefore, one cannot rely on a fixed pre-determined number of configuration types [7, 9, 11]; rather, one needs to be able to make as fine discriminations as required by the task. Objects participating in different actions may be very small, occupying only a few pixels in a low resolution image (brush, phone, Fig. 1a). In addition, these objects may be unknown a priori, such as in the natural case when the learning is weakly supervised, i.e. we know only the action label of the training images, while the participating objects are not annotated and cannot be independently learned as in [8, 11]. Finally, the background scene, used by [8, 11] to recognize sports actions and events, is uninformative for many transitive actions of interest, and cannot be directly utilized.

Since SAR is a version of an object recognition problem, a natural question to ask is whether it can be solved by directly applying state-of-the-art techniques of object recognition. As shown in the results section 3, the problem is significantly more difficult for current methods compared with more standard object recognition applications. The proposed method identifies more accurately the features and geometric relationships that contribute to correct recognition in this domain, leading to better recognition. It is further shown that integrating standard object recognition approaches into the proposed framework significantly improves their results in the SAR domain.

The main contribution of this paper is an approach, employing the so-called body anchored strategy explained below, for recognizing and distinguishing between similar transitive actions in single images. In both the learning and the test settings, the approach applies a two-stage interpretation to each (training or test) image. The first stage produces accurate detection and localization of body parts, and the second then extracts and uses features from locations anchored to body parts. In the implementation of the first stage, the face is detected first, and its detection is extended to accurately localize the elbow and the hand of the actor. In the second stage, the relative part locations and the hand region are analyzed for action related learning and recognition. During training, this allows the automatic discovery and construction of implicit non-parametric models for different important aspects of the actions, such as accurate relative part locations, relevant objects, types of grasping, and part-object configurations. During testing, this allows the approach to focus on image regions,

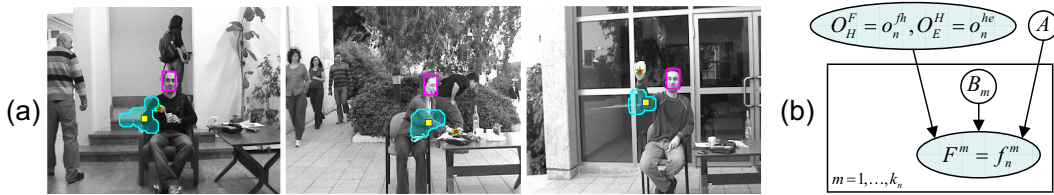

Figure 2: (a) Examples of the computed binary masks (cyan) for searching for elbow location given the detected hand and face marked by a red-green star and magenta rectangle respectively. The yellow square marks the detected elbow; (b) Graphical representation (in plate notation) of the proposed probabilistic model for action recognition (see section 2.2 for details).

features and relations that contain all of the relevant information for recognizing the action. As a result, we eliminate the need to have a priori models for the objects relevant for the action that were used in [11, 12]. Focusing in a body-anchored manner on the relevant information not only increases efficiency, but also considerably enhances recognition results. The approach is illustrated in fig. 1b.

The rest of the paper is organized as follows. Section 2 describes the proposed approach and its implementation details. Section 3 describes the experimental validation. Summary and discussion are provided in section 4.

## 2 Method

As outlined above, the approach proceeds in two main stages. The first stage is body interpretation, which is by itself a sequential process. First, the person is detected by detecting her/his face. Next, the face detection is extended to detect the hands and elbows of the person. This is achieved in a non-parametric manner by following chains of features connecting the face to the part of interest (hand, elbow), by an extension of [14]. In the second stage, features gathered from the hand region and the relative locations of the hand, face and elbow, are used to model and recognize the static action of interest. The first stage of the process, dealing with the face, hand and elbow detection, is described in section 2.1. The static action modeling and recognition is described in section 2.2 and additional implementation details are provided in section 2.3.

### 2.1 Body parts detection

Body parts detection in static images is a challenging problem, which has recently been addressed by several studies [14, 15, 16, 17, 18]. The most difficult parts to detect are the most flexible parts of the body - the lower arms and the hands. This is due to large pose and appearance variability and the small size typical to these parts. In our approach, we have adopted an extension of the non-parametric method for the detection of parts of deformable objects recently proposed by [14]. This method can operate in two modes. The first mode is used for the independent detection of sufficiently large and rigid objects and object parts, such as the face. The second mode allows propagating from some of the parts, which are independently detected, to additional parts, which are more difficult to detect independently, such as hands and elbows. The method extends the so-called star model by allowing features to vote for the detection target either directly, or indirectly, via features participating in feature-chains going towards the target. In the independent detection mode, these feature chains may start anywhere in the image, whereas in the propagation mode these chains must originate from already detected parts. The method employs a non-parametric generative probabilistic model, which can efficiently learn to detect any part from a collection of training sequences with marked target (e.g., hand) and source (e.g., face) parts (or only the target parts in the independent detection mode). The details of this model are described in [14]. In our approach, the face is detected in the independent detection mode of [14], and the hand and the elbow are detected by chains-propagation from the face detection (treated as the source part). The method is trained using a collection of short video sequences, each having the face, the hand and the elbow marked by three points. The code for the method of [14] was extended to allow restricted detection of dependent parts, such as hand and elbow. In some cases, the elbow is more difficult to detect than the hand, as it has less structure. For each (training or test) image $I_n$, we therefore constrain the elbow detection by a binary mask of possible elbow locations gathered from training images with

the sufficiently similar hand-face offset (within 0.25 face width) to the one detected on $I_n$. Figure 2a shows some examples of the detected faces, hands and elbows together with the elbow masks derived from the detected face-hand offset.

## 2.2 Modeling and recognition of static actions

Given an image $I_n$ (training or test), we first introduce the following notation (lower index refers to the image, upper indices to parts). Denote the instance of the action contained in $I_n$ by $a_n$ (known for training and unknown for test images). Denote the detected locations of the face by $x_n^f$, the hand by $x_n^h$, and the elbow by $x_n^e$. Also denote the width of the detected face by $s_n$. Throughout the paper, we will express all size and distance parameters in $s_n$ units, in order to eliminate the dependence on the scale of the person performing the action. For many transitive actions most of the discriminating information about the action resides in regions around specific body parts [19]. Here we focus on hand regions for hand-related actions, but for other actions their respective parts and part regions can be learned and used. We represent the information from the hand region by a set of rectangular patch features extracted from this region. All features are taken from a circular region with a radius $0.75 \cdot s_n$ around the hand location $x_n^h$. From this region we extract $s_n \times s_n$ pixel rectangular patch features centered at all Canny edge points sub-sampled with a $0.2 \cdot s_n$ pixel grid. Denote the set of patch features extracted from image $I_n$ by $\left\{ f_n^m = \left[ SIFT_n^m, \frac{1}{s_n} \left( x_n^m - x_n^f \right) \right] \right\}$, where $SIFT_n^m$ is the SIFT descriptor [20] of the $m$-th feature, $x_n^m$ is its image location, $\frac{1}{s_n} \left( x_n^m - x_n^f \right)$ is the offset (in $s_n$ units) between the feature and the face, and square brackets denote a row vector. The index $m$ enumerates the features in arbitrary order for each image. Denote by $k_n$ the number of patch features extracted from image $I_n$.

The probabilistic generative model explaining all the gathered data is defined as follows. The observed variables of the model are: the face-hand offset $O_H^F = o_n^{fh} \equiv x_n^h - x_n^f$, the hand-elbow offset $O_E^H = o_n^{he} \equiv x_n^e - x_n^h$, and the patch features $\{F^m = f_n^m\}$. The unobserved variables of the model are the action label variable $A$, and the set of binary variables $\{B^m\}$, one for each extracted patch feature. The meaning of $B^m = 1$ is that the $m$-th patch feature was generated by the action $A$, while the meaning of $B^m = 0$ is that the $m$-th patch feature was generated independently of $A$. Throughout the paper we will use a shorthand form of variable assignments, e.g., $P\left( o_n^{fh}, o_n^{he} \right)$ instead of $P\left( O_H^F = o_n^{fh}, O_E^H = o_n^{he} \right)$. We define the joint distribution of the model that generates the data for image $I_n$ as:

$$P\left( A, \{B^m\}, o_n^{fh}, o_n^{he}, \{f_n^m\} \right) = P\left(A\right) \cdot P\left( o_n^{fh}, o_n^{he} \right) \cdot \prod_{m=1}^{k_n} P\left( B^m \right) \cdot P\left( f_n^m \,\middle|\, A, o_n^{fh}, o_n^{he}, B^m \right)$$

(1)

Here $P\left(A\right)$ is a prior action distribution, which we take to be uniform, and:

$$P\left( f_n^m \,\middle|\, A, o_n^{fh}, o_n^{he}, B^m \right) = \left\{ \begin{array}{ll} P\left( f_n^m \,\middle|\, A, o_n^{fh}, o_n^{he} \right) & \text{if } B^m = 1 \\ P\left( f_n^m \,\middle|\, o_n^{fh}, o_n^{he} \right) & \text{otherwise} \end{array} \right.$$

(2)

The $P\left( B^m \right) = \alpha$, is the prior probability for the $m$-th feature to be generated from the action, and we assume it maintains the following relation: $P\left( B^m = 1 \right) = \alpha \ll (1 - \alpha) = P\left( B^m = 0 \right)$ reflecting the fact that most patch features are not related to the action. Figure 2b shows the graphical representation of the proposed model.

As shown in the Appendix A, in order to find the action label assignment to $A$ that maximizes the posterior of the proposed probabilistic generative model, it is sufficient to compute:

$$\arg\max_A \log P\left( A \,\middle|\, o_n^{fh}, o_n^{he}, \{f_n^m\} \right) = \arg\max_A \sum_{m=1}^{k_n} \frac{P\left( f_n^m, A, o_n^{fh}, o_n^{he} \right)}{P\left( f_n^m, o_n^{fh}, o_n^{he} \right)}$$

(3)

As can be seen from eq. 3, and as shown in the Appendix A, the inference is independent of the exact value of $\alpha$ (as long as $\alpha \ll (1 - \alpha)$). In section 2.3 we explain how to empirically estimate the probabilities $P\left( f_n^m, A, o_n^{fh}, o_n^{he} \right)$ and $P\left( f_n^m, o_n^{fh}, o_n^{he} \right)$ that are necessary to compute 3.

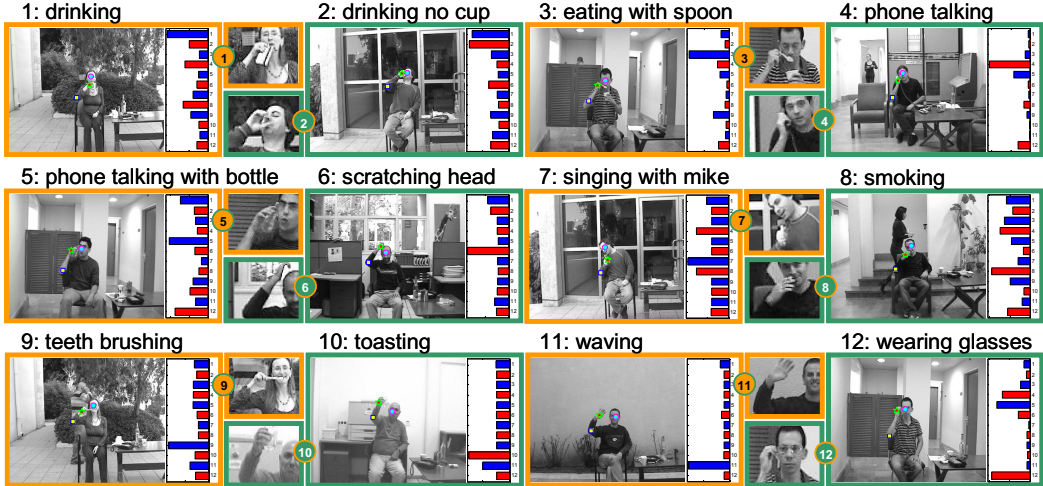

Figure 3: Examples of similar static transitive action recognition on our 12-actions / 10-people ('12/10') dataset. On all examples, the detected face, hand and elbow are shown by cyan circle, red-green star and yellow square, respectively. At the right hand side of each image, the bar graph shows the estimated log-posterior of the action variable $A$. Each example shows a zoomed-in ROI of the action. Additional examples are provided in supplementary material.

## 2.3   Model probabilities

The model probabilities are estimated from the training data using Kernel Density Estimation (KDE) [21]. Assume we are given a set of samples $\{Y_1, \ldots, Y_R\}$ from some distribution of interest. Given a new sample $Y$ from the same distribution, a symmetric Gaussian KDE estimate $P(Y)$ for the probability of $Y$ can be approximated as: $P(Y) \approx \frac{1}{R} \cdot \sum_{Y_r \in NN(Y)} \exp\left(-0.5 \cdot \|Y - Y_r\|^2 \big/ \sigma^2\right)$ where $NN(Y)$ is the set of nearest neighbors of $Y$ within the given set of samples. When the number of samples $R$ is large, brute-force search for the $NN(Y)$ set becomes infeasible. Therefore, we use Approximate Nearest Neighbor (ANN) search (using the implementation of [22]) to compute the KDE. To compute $P\left(f_n^m, A = a, o_n^{fh}, o_n^{he}\right)$ for the $m$-th patch feature $f_n^m = \left[SIFT_n^m, \frac{1}{s_n}\left(x_n^m - x_n^h\right)\right]$ in test image $I_n$, we search for the nearest neighbors of the row vector $\left[f_n^m, \frac{1}{s_n} o_n^{fh}, \frac{1}{s_n} o_n^{he}\right]$ in a set of row vectors: $\left\{\left[f_t^r, \frac{1}{s_t} o_t^{fh}, \frac{1}{s_t} o_t^{he}\right]\middle|\text{ all } f_t^r \text{ in training images } I_t, \text{ s.t. } a_t = a\right\}$ using an ANN query. Recall that $s_n$ was defined as the width of the detected face in image $I_n$, and hence $\frac{1}{s_n}$ is the scale factor that we use for the offsets in the query. The query returns a set of $K$ nearest neighbors, and the Gaussian KDE with $\sigma = 0.2$, is applied to this set to compute the estimated probability $P\left(f_n^m, A = a, o_n^{fh}, o_n^{he}\right)$. In our experiments we found that it is sufficient to use $K = 25$. The $P\left(f_n^m, o_n^{fh}, o_n^{he}\right)$ is computed as: $P\left(f_n^m, o_n^{fh}, o_n^{he}\right) = \sum_a P\left(f_n^m, A = a, o_n^{fh}, o_n^{he}\right)$.

## 3   Results

To test our approach, we have applied it to two static transitive action recognition datasets. The first dataset, denoted '12/10' dataset, was created by us and contained 12 similar transitive actions performed by 10 different people, appearing against different natural backgrounds. The second dataset was compiled by [11] for dynamic transitive action recognition. It contains 9 different people performing 6 general transitive actions. Although originally designed and used by Gupta *et al* in [11] for dynamic action recognition, we transformed it into a static action recognition dataset by assigning action labels to frames actually containing the actions and treating each such frame as a separate static instance of the action. Since successive frames are not independent, the experiments conducted on both datasets were all performed in a person-leave-one-out manner, meaning that during the training we completely excluded all the frames of the tested person. Section 3.1 provides more details on the relevant parts (face, hand, and elbow) detection in our experiments complementing section 2.1. Sections 3.2 and 3.3 describe the '12/10' and the Gupta *et al* datasets in more detail

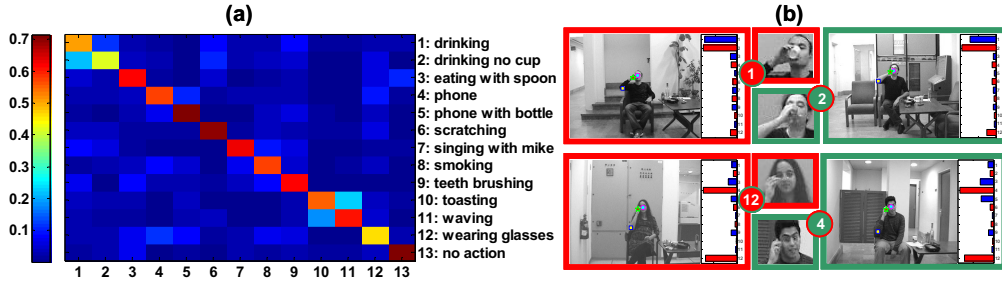

Figure 4: (a) Average static action confusion matrix obtained by leave-one-out cross validation of the proposed method on the '12/10' dataset; (b) Some interesting failures (red boxes), on the right of each failure there is a successfully recognized instance of an action with which the method has confused. The meaning of the bar-graph is as in figure 3. Additional failure examples are provided in the supplementary material.

together with the respective static action recognition experiments performed on them. All experiments were performed on grayscale versions of the images. Figures 3 and 6a illustrate the two tested datasets also showing examples of successfully recognized static transitive actions, and figure 4b shows some interesting failures.

## 3.1 Part detection details

Our approach is based on prior part detection and its performance is bounded from above by the part detection performance. The detection rates of state-of-the-art methods for localizing body parts in a general setting are currently a significant limiting factor. For example, [14] that we use here, obtains an average of 66% correct hand detection (comparing favorably to other state-of-the-art methods) in the general setting experiments, when both the person and the background are unseen during part detector training. However, as shown in [14], average 85% part detection performance can be achieved in more restricted settings. One such setting (denoted self-trained) is when an independent short part detection training period of several seconds is allowed for each test person, as for e.g. in the human-computer interaction applications. Another setting (denoted environment-trained) is when the environment in which people perform the action is fixed, e.g. in applications where we can train part detectors on some people, and then apply them to new unseen people, but appearing in the same environment. As demonstrated in the methods comparison experiment in section 3.2, it appears that part detection is an essential component of solving SAR. Current performance in automatic body parts detection is well below human performance, but the area is now a focus of active research which is likely to reduce this current performance gap. In our experiments we adopted the more constrained (but still useful) part detection settings described above, the self-trained for the 12-10 dataset (having each person in different environment) and the environment-trained for the Gupta et al. dataset (having all the people in the same general environment).

In the 12-10 dataset experiments, the part detection models for the face, hand and elbow described in section 2.1, were trained using 10 additional short movies, one for each person, in which the actors randomly moved their hands. On these 10 movies, face, hand and elbow locations were manually marked. The learned models were then applied to detect their respective parts on the 120 movie sequences of our dataset. The hand detection performance was 94.1% (manually evaluated on a subset of frames). Qualitative examples are provided in the supplementary material. The part detection for the Gupta et al. dataset was performed in a person-leave-one-out manner. For each person the parts (face, hand) were detected using models trained on other people. The mean hand detection performance was 88% (manually evaluated). Since most people in the dataset wear very dark clothing, in many cases the elbow is invisible and therefore it was not used in this experiment (it is straightforward to remove it from the model by assigning a fixed value to the hand-elbow offset in both training and test).

## 3.2 The '12/10' dataset experiments

The '12/10' dataset consists of 120 videos of 10 people performing 12 similar transitive actions, namely drinking, talking on the phone, scratching, toasting, waving, brushing teeth, smoking, wearing glasses, eating with a spoon, singing to a microphone, and also drinking without a cup and

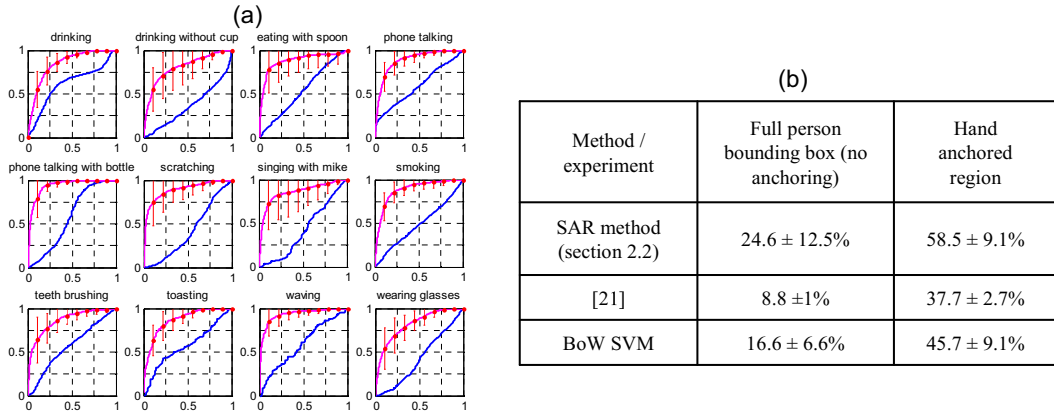

| Method / experiment | Full person bounding box (no anchoring) | Hand anchored region |
|---|---|---|
| SAR method (section 2.2) | 24.6 ± 12.5% | 58.5 ± 9.1% |
| [21] | 8.8 ±1% | 37.7 ± 2.7% |
| BoW SVM | 16.6 ± 6.6% | 45.7 ± 9.1% |

Figure 5: (a) ROC based comparison with the state-of-the-art method of object detection [23] applied to recognize static actions. For each action, the blue line is the average ROC of [23], and the magenta line is the average ROC of the proposed method. (b) Comparing state-of-the-art object recognition methods on the SAR task with and without 'body anchoring'.

making a phone call with a bottle. All people except one were filmed against different backgrounds. All backgrounds were natural indoor / outdoor scenes containing both clutter and people passing by. The drinking and toasting actions were performed with 2-4 different tools, and phone talking was performed with mobile and regular phones. Overall, the dataset contains 44,522 frames. Not all frames contain actions of interest (e.g. in drinking there are frames where the person reaches to / puts down a cup). The ground-truth action labels were manually assigned to the relevant frames. Each of the resulting 23,277 relevant action frames was considered a separate instance of an action. The remaining frames were labeled 'no-action'. The average recognition accuracy was $59.3\pm8.6$ for the 13 actions (including no-action) and $58.5 \pm 9.1\%$ for the 12 main actions (excluding no-action). Figure 4a shows the confusion matrix for 13 actions averaged over the 10 test people.

As mentioned in the introduction, one of the important questions we wish to answer is the need for the detection of the fine details of the person, such as the accurate hand, and elbow locations, in order to recognize the action. To test this issue, we have compared the results of three approaches: deformable parts model [23], Bag-of-Words (BoW) SVM ([24]), and our approach described in section 2.2, in two settings. In the first setting the methods were trained to distinguish between the actions based on a bounding box of the entire person (i.e. without focusing on the fine details such as provided by the hand and elbow detection). In the second, body anchored setting, the methods were applied to the hand anchored regions (small regions around the detected hand as described in section 2.2). The method of [23] is one of the state-of-the-art object recognition schemes, achieving top scores on recent PASCAL-VOC competitions [25], and BoW SVM is a popular method in the literature also obtaining state-of-the art results for some datasets. Figure 5b shows the results obtained by the three methods in the two settings. Figure 5a provides ROC-based comparison of the results of our full approach with the ones obtained by [23]. The obtained results strongly suggest that body anchoring is a powerful prior for the task of distinguishing between similar transitive actions.

### 3.3 Gupta *et al* dataset experiments

This dataset was compiled by [11]. It consists of 46 movie sequences of 9 different people performing 6 distinct transitive actions: drinking, spraying, answering the phone, making a call, pouring from a pitcher and lighting a flashlight. In each movie, we manually assigned action labels to all the frames actually containing the action, labeling the remainder of the frames 'no-action'. Since the distinction between 'making a call' and 'answering phone' was in the presence or absence of the 'dialing' action in the respective video, we re-labeled the frames of these actions into 'phone talking' and 'dialing'. The action recognition performance was measured using the person-leave-one-out cross-validation, in the same manner as for our dataset. The average accuracy over the 7 static actions (including no-action) was $82 \pm 11.5\%$, and was $86 \pm 14.4\%$ excluding no-action. The average 7-action confusion matrix is shown in figure 6b. The presented results are for the static action recognition, and hence are not directly comparable with the results obtained on this dataset for the dynamic action recognition by [11], who obtained $93.34\%$ recognition (out of the 46 video

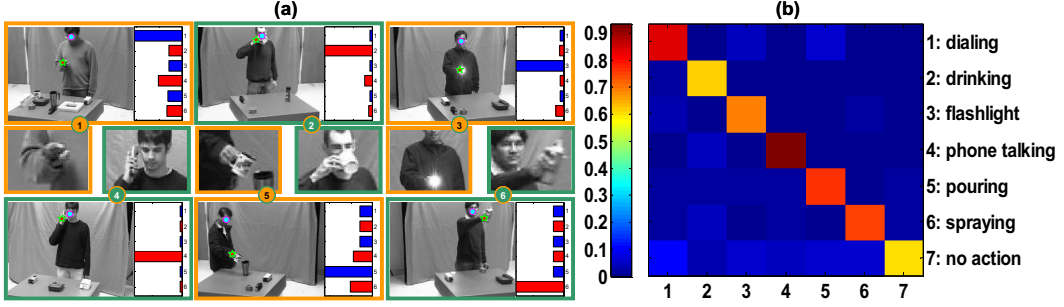

Figure 6: (a) some successfully identified action examples from the dataset of [11]; (b) mean static action confusion matrix for leave-one-out cross validation experiments on the Gupta et al. dataset.

sequences and not frames) using both the temporal information (part tracks, etc.) and a priori models for the participating objects (cup, pitcher, flashlight, spray bottle and phone).

## 4   Discussion

We have presented a method for recognizing transitive actions from single images. This task is performed naturally and efficiently by humans, but performance by current recognition methods is severely limited. The proposed method can successfully handle both similar transitive actions (the '12/10' dataset), and general transitive actions (the Gupta et al dataset). The method uses priors that focus on body part anchored features and relations. It has been shown that most common verbs are associated with specific body parts [19]; the actions considered here were all hand-related in this sense. The detection of hands and elbows therefore provided useful priors in terms of regions and properties likely to contribute to the SAR task in this setting. The proposed approach can be generalized to deal with other actions by detecting all the body parts associated with common verbs, automatically detecting the relevant parts for each specific action during training, and finally applying the body anchored SAR model described in section 2.2. The comparisons show that without using the body anchored priors there is a highly significant drop in SAR performance even when employing state-of-the-art methods for object recognition. The main reasons for this drop are the fine details and local nature of the relevant evidence for distinguishing between actions, the huge number of possible locations, and detailed features that need to be searched if body-anchored priors are not used. Directions for future studies therefore include a more complete and accurate body parts detection and their use in providing useful priors for static action recognition and interpretation.

## A   Log-posterior derivation

Here we derive the equivalent form of log-posterior (eq. 3) of the proposed probabilistic action recognition model defined in eq. 1. In 4, the symbol $\sim$ means equivalent in terms of maximizing over the values of the action variable $A$.

$$
\begin{aligned}
&\log P\left(A \,\middle|\, o_n^{fh}, o_n^{he}, \{f_n^m\}\right) \sim \log \sum_{\{B^m\}} P\left(A, \{B^m\}, o_n^{fh}, o_n^{he}, \{f_n^m\}\right) = \\
&\log \left[ P\left(A\right) \cdot P\left(o_n^{fh}, o_n^{he}\right) \cdot \prod_{m=1}^{k_n} \left[ \sum_{B^m=0}^{1} P\left(B^m\right) \cdot P\left(f_n^m \,\middle|\, A, o_n^{fh}, o_n^{he}, B^m\right) \right] \right] \sim \\
&\sum_{m=1}^{k_n} \log \left[ \sum_{B^m=0}^{1} P\left(B^m\right) \cdot P\left(f_n^m \,\middle|\, A, o_n^{fh}, o_n^{he}, B^m\right) \right] = \\
&\sum_{m=1}^{k_n} \log \left[ \alpha \cdot P\left(f_n^m \,\middle|\, A, o_n^{fh}, o_n^{he}\right) + (1-\alpha) \cdot P\left(f_n^m \,\middle|\, o_n^{fh}, o_n^{he}\right) \right] = \\
&\sum_{m=1}^{k_n} \log \left[ 1 + \frac{P\left(f_n^m \,\middle|\, A, o_n^{fh}, o_n^{he}\right)}{\gamma \cdot P\left(f_n^m \,\middle|\, o_n^{fh}, o_n^{he}\right)} \right] + \sum_{m=1}^{k_n} \log \left[ (1-\alpha) \cdot P\left(f_n^m \,\middle|\, o_n^{fh}, o_n^{he}\right) \right] \sim \\
&\sum_{m=1}^{k_n} \log \left[ 1 + \frac{P\left(f_n^m \,\middle|\, A, o_n^{fh}, o_n^{he}\right)}{\gamma \cdot P\left(f_n^m \,\middle|\, o_n^{fh}, o_n^{he}\right)} \right] \underset{(*)}{\approx} \sum_{m=1}^{k_n} \frac{P\left(f_n^m \,\middle|\, A, o_n^{fh}, o_n^{he}\right)}{\gamma \cdot P\left(f_n^m \,\middle|\, o_n^{fh}, o_n^{he}\right)} \sim \sum_{m=1}^{k_n} \frac{P\left(f_n^m \,\middle|\, A, o_n^{fh}, o_n^{he}\right)}{P\left(f_n^m \,\middle|\, o_n^{fh}, o_n^{he}\right)} \sim \\
&\sum_{m=1}^{k_n} \frac{P\left(f_n^m, A, o_n^{fh}, o_n^{he}\right)}{P\left(f_n^m, o_n^{fh}, o_n^{he}\right)}
\end{aligned}
$$

$$(4)$$

In eq. 4, $\gamma = \alpha / (1 - \alpha)$, the term $\sum_{m=1}^{k_n} \log \left[ (1-\alpha) \cdot P\left(f_n^m \,\middle|\, o_n^{fh}, o_n^{he}\right) \right]$ is independent of the action (constant for a given image $I_n$) and thus can be dropped, and $(*)$ follows from $\log(1+\varepsilon) \approx \varepsilon$ for $\varepsilon \ll 1$ and from $\gamma$ being large due to our assumption that $\alpha \ll (1-\alpha)$.

# References

[1] Jackendoff, R.: Semantic interpretation in generative grammar. The MIT Press (1972)

[2] Laptev, I., Marszalek, M., Schmid, C., Rozenfeld, B.: Learning realistic human actions from movies. In: CVPR. (2008) 1–8

[3] Iacoboni, M., Mazziotta, J.C.: Mirror neuron system: basic findings and clinical applications. Annals of Neurology (2007)

[4] Kim, J., Biederman, I.: Where do objects become scenes? Journal of Vision (2009)

[5] Helbig, H., Graf, M., Kiefer, M.: The role of action representations in visual object recognition. Experimental Brain Research (2006)

[6] Sakata, H., Taira, M., Kusunoki, M., Murata, A., Tanaka, Y., Tsutsui, K.: Neural coding of 3d features of objects for hand action in the parietal cortex of the monkey. Philos Trans R Soc Lond B Biol Sci. (1998)

[7] Wang, Y., Jiang, H., Drew, M.S., nian Li, Z., Mori, G.: Unsupervised discovery of action classes. In: CVPR. (2006) 5

[8] Li, L., Fei-Fei, L.: What, where and who? classifying events by scene and object recognition. In: ICCV. (2007) 1–8

[9] Thurau, C., Hlavac, V.: Pose primitive based human action recognition in videos or still images. In: CVPR. (2008) 1–8

[10] Yao, B., Fei-Fei, L.: Grouplet: A structured image representation for recognizing human and object interactions. CVPR (2010)

[11] Gupta, A., Kembhavi, A., Davis, L.: Observing human-object interactions: Using spatial and functional compatibility for recognition. PAMI (2009)

[12] Yao, B., Fei-Fei, L.: Modeling mutual context of object and human pose in human-object interaction activities. CVPR (2010)

[13] Blake, A., Rother, C., Brown, M., Perez, P., Torr, P.: Interactive image segmentation using an adaptive gmmrf model. ECCV (2004)

[14] Karlinsky, L., Dinerstein, M., Harari, D., Ullman, S.: The chains model for detecting parts by their context. CVPR (2010)

[15] Ferrari, V., Marin, M., Zisserman, A.: Progressive search space reduction for human pose estimation. CVPR (2008)

[16] Andriluka, M., Roth, S., Schiele, B.: Pictorial structures revisited: People detection and articulated pose estimation. CVPR (2009)

[17] Felzenszwalb, P., Huttenlocher, D.: Pictorial structures for object recognition. IJCV **61** (2005) 55–79

[18] Ramanan, D., Forsyth, D.A., Barnard, K.: Building models of animals from video. PAMI (2006)

[19] Maouene, J., Hidaka, S., Smith, L.B.: Body parts and early-learned verbs. Cognitive Science (2008)

[20] Lowe, D.: Distinctive image features from scale-invariant keypoints. IJCV (2004)

[21] Duda, R., Hart, P.: Pattern classification and scene analysis. Wiley (1973)

[22] Mount, D., Arya, S.: Ann: A library for approximate nearest neighbor searching. CGC 2nd Annual Workshop on Comp. Geometry (1997)

[23] Felzenszwalb, P., McAllester, D., Ramanan, D.: A discriminatively trained, multiscale, deformable part model. CVPR (2008) 1–8

[24] Zhang, J., Marszalek, M., Lazebnik, S., Schmid, C.: Local features and kernels for classification of texture and object categories: A comprehensive study. IJCV (2007)

[25] Everingham, M., Van Gool, L., Williams, C., Winn, J., Zisserman, A.: The pascal visual object classes challenge 2007 results. http://pascallin.ecs.soton.ac.uk/challenges/VOC/voc2007 (2007)

